# On the Analysis of Multi-Channel Neural Spike Data

**Bo Chen, David E. Carlson and Lawrence Carin**
Department of Electrical and Computer Engineering, Duke University, Durham, NC 27708
{bc69, dec18, lcarin}@duke.edu

## Abstract

Nonparametric Bayesian methods are developed for analysis of *multi-channel* spike-train data, with the feature learning and spike sorting performed jointly. The feature learning and sorting are performed simultaneously across all channels. Dictionary learning is implemented via the beta-Bernoulli process, with spike sorting performed via the dynamic hierarchical Dirichlet process (dHDP), with these two models coupled. The dHDP is augmented to eliminate refractory-period violations, it allows the "appearance" and "disappearance" of neurons over time, and it models smooth variation in the spike statistics.

## 1  Introduction

The analysis of action potentials ("spikes") from neural-recording devices is a problem of long-standing interest (see [21, 1, 16, 22, 8, 4, 6] and the references therein). In such research one is typically interested in clustering (sorting) the spikes, with the goal of linking a given cluster to a particular neuron. Such technology is of interest for brain-machine interfaces and for gaining insight into the properties of neural circuits [14]. In such research one typically ($i$) filters the raw sensor readings, ($ii$) performs thresholding to "detect" the spikes, ($iii$) maps each detected spike to a feature vector, and ($iv$) then clusters the feature vectors [12]. Principal component analysis (PCA) is a popular choice [12] for feature mapping. After performing such sorting, one typically must ($v$) search for refractory-time violations [5], which occur when two or more spikes that are sufficiently proximate are improperly associated with the same cluster/neuron (which is impossible due to the refractory time delay required for the same neuron to re-emit a spike). Recent research has combined ($iii$) and ($iv$) within a single model [6], and methods have been developed recently to address ($v$) while performing ($iv$) [5].

Many of the early methods for spike sorting were based on classical clustering techniques [12] (*e.g.*, K-means and GMMs, with a fixed number of mixtures), but recently Bayesian methods have been developed to account for more modeling sophistication. For example, in [5] the authors employed a modification to the Chinese restaurant formulation of the Dirichlet process (DP) [3] to automatically infer the number of clusters (neurons) present, allow statistical drift in the feature statistics, permit the "appearance"/"disappearance" of neurons with time, and automatically account for refractory-time requirements within the clustering (*not* as a post-clustering step). However, [5] assumed that the spike features were provided via PCA in the first two or three principal components (PCs). In [6] feature learning and spike sorting were performed jointly via a mixture of factor analyzers (MFA) formulation. However, in [6] model selection was performed (for the number of features and number of neurons) and a maximum likelihood (ML) "point" estimate was constituted for the model parameters; since a fixed number of clusters are inferred in [6], the model does not directly allow for the "appearance"/"disappearance" of neurons, or for any temporal dependence to the spike statistics.

There has been an increasing interest in developing neural devices with $C > 1$ recording channels, each of which produces a separate electrical recording of neural activity. Recent research shows increased system performance with large $C$ [18]. Almost all of the above research on spike sorting

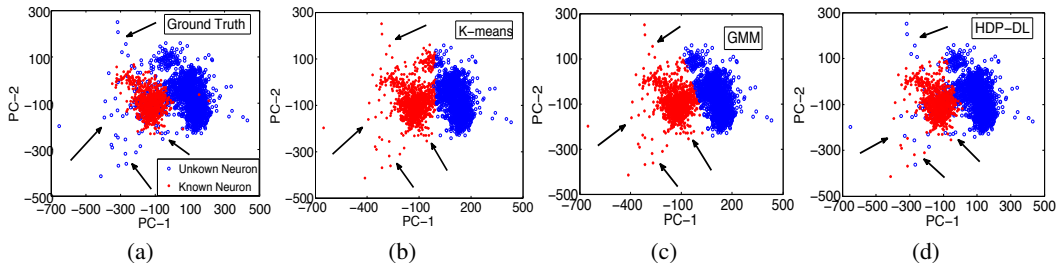

Figure 1: *Comparison of spike sorting on real data. (a) Ground truth; (b) K-means clustering on the first 2 principal components; (c) GMM clustering with the first 2 principal components; (d) proposed method. We label using arrows examples K-means and the GMM miss, and that the proposed method properly sort.*

has been performed on a single channel, or when multiple channels are present each is typically analyzed in isolation. In [5] $C = 4$ channels were considered, but it was assumed that a spike occurred at the same time (or nearly same time) across all channels, and the features from the four channels were concatenated, effectively reducing this again to a single-channel analysis. When $C \gg 1$, the assumption that a given neuron is observed simultaneously on all channels is typically inappropriate, and in fact the diversity of neuron sensing across the device is desired, to enhance functionality [18].

This paper addresses the multi-channel neural-recording problem, under conditions for which concatenation may be inappropriate; the proposed model generalizes the DP formulation of [5], with a *hierarchical* DP (HDP) formulation [20]. In this formulation statistical strength is shared across the channels, without assuming that a given neuron is simultaneously viewed across all channels. Further, the model generalizes the HDP, via a *dynamic* HDP (dHDP) [17] to allow the "appearance"/"disappearance" of neurons, while also allowing smooth changes in the statistics of the neurons. Further, we explicitly account for refractory times, as in [5]. We also perform joint feature learning and clustering, using a mixture of factor analyzers construction as in [6], but we do so in a fully Bayesian, multi-channel setting (additionally, [6] did not account for time-varying statistics). The learned factor loadings are found to be similar to wavelets, but they are matched to the properties of neuron spikes; this is in contrast to previous feature extraction on spikes [11] based on orthogonal wavelets, that are not necessarily matched to neuron properties.

To give a preview of the results, providing a sense of the importance of feature learning (relative to mapping data into PCA features learned offline), in Figure 1 we show a comparison of clustering results on the first channel of d533101 data from hc-1 [7]. For all cases in Figure 1 the data are *depicted* in the first two PCs *for visualization*, but the proposed method in (d) *learns* the number of features and their composition, while simultaneously performing clustering. The results in (b) and (c) correspond respectively to widely employed K-means and GMM analysis, based on using two PCs (in these cases the analysis *are* employed in PCA space, as have been many more-advanced approaches [5]). From Figures 1 (b) and (c), we observe that both K-means and GMM work well, but due to the constrained feature space they incorrectly classify some spikes (marked by arrows). However, the proposed model, shown in Figure 1(d), which incorporates dictionary learning with spike sorting, infers an appropriate feature space (not shown) and more effectively clusters the neurons. The details of this model, including a multi-channel extension, are discussed in detail below.

## 2 Model Construction

### 2.1 Dictionary learning

We initially assume that spike detection has been performed on all channels. Spike $n \in \{1, \ldots, N_c\}$ on channel $c \in \{1, \ldots, C\}$ is a vector $\boldsymbol{x}_n^{(c)} \in \mathbb{R}^D$, defined by $D$ time samples for each spike, centered at the peak of the detected signal; there are $N_c$ spikes on channel $c$.

Data from spike $n$ on channel $c$, $\boldsymbol{x}_n^{(c)}$, is represented in terms of a dictionary $\mathbf{D} \in \mathbb{R}^{D \times K}$, where $K$ is an upper bound on the number of needed dictionary elements (columns of $\mathbf{D}$), and the model

infers the subset of dictionary elements needed to represent the data. Each $\boldsymbol{x}_n^{(c)}$ is represented as

$$\boldsymbol{x}_n^{(c)} = \mathbf{D}\boldsymbol{\Lambda}^{(c)}\boldsymbol{s}_n^{(c)} + \boldsymbol{\epsilon}_n^{(c)} \tag{1}$$

where $\boldsymbol{\Lambda}^{(c)} = \mathrm{diag}(\lambda_1^{(c)}b_1, \lambda_2^{(c)}b_2, \ldots, \lambda_K^{(c)}b_K)$ is a diagonal matrix, with $\boldsymbol{b} = (b_1, \ldots, b_K)^T \in \{0,1\}^K$. Defining $\boldsymbol{d}_k$ as the $k$th column of $\mathbf{D}$, and letting $\mathbf{I}_D$ represent the $D \times D$ identity matrix, the priors on the model parameters are

$$\boldsymbol{d}_k \sim \mathcal{N}(\mathbf{0}, \frac{1}{D}\mathbf{I}_D)\,, \qquad \lambda_k^{(c)} \sim \mathcal{TN}^+(0, \gamma_c^{-1})\,, \qquad \boldsymbol{\epsilon}_n^{(c)} \sim \mathcal{N}(\mathbf{0}, \boldsymbol{\Sigma}_c^{-1}) \tag{2}$$

where $\boldsymbol{\Sigma}_c = \mathrm{diag}(\eta_1^{(c)}, \ldots, \eta_D^{(c)})$, and $\mathcal{TN}^+(\cdot)$ represents the truncated (positive) normal distribution. Gamma priors (detailed when presenting results) are placed on $\gamma_c$ and on each of the elements of $(\eta_1^{(c)}, \ldots, \eta_D^{(c)})$. For the binary vector $\boldsymbol{b}$ we impose the prior $b_k \sim \mathrm{Bernoulli}(\pi_k)$, with $\pi_k \sim \mathrm{Beta}(a/K, b(K-1)/K)$, implying that the number of non-zero components of $\boldsymbol{b}$ is drawn $\mathrm{Binomial}(K, a/(a+b(K-1)))$; this corresponds to $\mathrm{Poisson}(a/b)$ in the limit $K \to \infty$. Parameters $a$ and $b$ are set to favor a sparse $\boldsymbol{b}$.

This model imposes that each $\boldsymbol{x}_n^{(c)}$ is drawn from a linear subspace, defined by the columns of $\mathbf{D}$ with corresponding non-zero components in $\boldsymbol{b}$; the same linear subspace is shared across all channels $c \in \{1, \ldots, C\}$. However, the strength with which a column of $\mathbf{D}$ contributes toward $\boldsymbol{x}_n^{(c)}$ depends on the channel $c$, as defined by $\boldsymbol{\Lambda}^{(c)}$. Concerning $\boldsymbol{\Lambda}^{(c)}$, rather than explicitly imposing a sparse diagonal via $\boldsymbol{b}$, we may also draw $\lambda_k^{(c)} \sim \mathcal{TN}^+(0, \gamma_{ck}^{-1})$, with shrinkage priors employed on the $\gamma_{ck}$ (i.e., with the $\gamma_{ck}$ drawn from a gamma prior that favors large $\gamma_{ck}$; which encourages many of the diagonal elements of $\boldsymbol{\Lambda}^{(c)}$ to be small, but typically not exactly zero). In tests, the model performed similarly when shrinkage priors were used on $\boldsymbol{\Lambda}^{(c)}$ relative to explicit imposition of sparseness via $\boldsymbol{b}$; all results below are based on the latter construction.

## 2.2 Multi-Channel Dynamic hierarchical Dirichlet process

We sort the spikes on the channels by clustering the $\{\boldsymbol{s}_n^{(c)}\}$, and in this sense feature design (learning $\{\mathbf{D}\boldsymbol{\Lambda}^{(c)}\}$) and sorting are performed simultaneously. We first discuss how this may be performed via a hierarchical Dirichlet process (HDP) construction [20], and then extend this via a dynamic HDP (dHDP) [17] considering multiple channels. In an HDP construction, the $\{\boldsymbol{s}_n^{(c)}\}$ are modeled as being drawn

$$\boldsymbol{s}_n^{(c)} \sim f(\theta_n^{(c)})\,, \qquad \theta_n^{(c)} \sim G^{(c)}\,, \qquad G^{(c)} \sim \mathrm{DP}(\alpha_c G)\,, \qquad G \sim \mathrm{DP}(\alpha_0 G_0) \tag{3}$$

where a draw from, for example, $\mathrm{DP}(\alpha_0 G_0)$ may be constructed [19] as $G = \sum_{i=1}^{\infty} \pi_i \delta_{\theta_i^*}$, where $\pi_i = V_i \prod_{h<i}(1 - V_h)$, $V_i \sim \mathrm{Beta}(1, \alpha_0)$, $\theta_i^* \sim G_0$, and $\delta_{\theta_i^*}$ is a unit point measure situated at $\theta_i^*$. Each of the $G^{(c)}$ is therefore of the form $G^{(c)} = \sum_{i=1}^{\infty} \pi_i^{(c)} \delta_{\theta_i^*}$, with $\sum_{i=1}^{\infty} \pi_i^{(c)} = 1$ and with the $\{\theta_i^*\}$ shared across all $G^{(c)}$, but with channel-dependent ($c$-dependent) probability of using elements of $\{\theta_i^*\}$. Gamma hyperpriors are employed for $\{\alpha_c\}$ and $\alpha_0$. In the context of the model developed in Section 2.1, the density function $f(\cdot)$ corresponds to a Gaussian, and parameters $\theta_i^* = (\boldsymbol{\mu}_i^*, \Gamma_i^*)$ correspond to means and precision matrices, with $G_0$ a normal-Wishart distribution. The proposed model may be viewed as an mixture of factor analyzers (MFA) [6] applied to each channel, with the addition of sharing of statistical strength across the $C$ channels via the HDP. Sharing is manifested in two forms: ($i$) via the shared linear subspace defined by the columns of $\mathbf{D}$, and ($ii$) via hierarchical clustering via HDP of the relative weightings $\{\boldsymbol{s}_n^{(c)}\}$. In tests, the use of channel-dependent $\boldsymbol{\Lambda}^{(c)}$ was found critical to modeling success, as compared to employing a single $\boldsymbol{\Lambda}$ shared across all channels.

The above HDP construction assumes that $G^{(c)} = \sum_{i=1}^{\infty} \pi_i^{(c)} \delta_{\theta_i^*}$ is time-independent, implying that the probability $\pi_i^{(c)}$ that $\boldsymbol{x}_n^{(c)}$ is drawn from $f(\theta_i^*)$ is time invariant. There are two ways this assumption may be violated. First, neuron refractory time implies a minimum delay between consecutive firing of the same neuron; this effect is addressed in a relatively straightforward manner discussed in Section 2.3. The second issue corresponds to the "appearance" or "disappearance" of neurons [5]; the former would be characterized by an increase in the value of a component of $\pi_i^{(c)}$, while the latter would be characterized by one of the components of $\pi_i^{(c)}$ going to zero (or near zero). It is

desirable to augment the model to address these objectives. We achieve this by application of the dHDP construction developed in [17].

As in [5], we divide the time axis into contiguous, non-overlapping temporal blocks, where block $j$ corresponds to spikes observed between times $\tau_{j-1}$ and $\tau_j$; we consider $J$ such blocks, indexed $j = 1, \ldots, J$. The spikes on channel $c$ within block $j$ are denoted $\{x_{jn}^{(c)}\}_{n=1,N_{cj}}$, where $N_{cj}$ represents the number of spikes within block $j$ on channel $c$. In the dHDP construction we have

$$s_{jn}^{(c)} \sim f(\theta_{jn}^{(c)}), \quad \theta_{jn}^{(c)} \sim w_j^{(c)} G_j^{(c)} + (1 - w_j^{(c)}) G_{j-1}^{(c)} \tag{4}$$

$$G_j^{(c)} \sim \text{DP}(\alpha_{jc} G), \quad G \sim \text{DP}(\alpha_0 G_0), \quad w_j^{(c)} \sim \text{Beta}(c, d) \tag{5}$$

where $w_1^{(c)} = 1$ for all $c$. The expression $w_j^{(c)}$ controls the probability that $\theta_{jn}^{(c)}$ is drawn from $G_j^{(c)}$, while with probability $1 - w_j^{(c)}$ parameter $\theta_{jn}^{(c)}$ is drawn from $G_{j-1}^{(c)}$. The cumulative mixture model $w_j^{(c)} G_j^{(c)} + (1 - w_j^{(c)}) G_{j-1}^{(c)}$ supports arbitrary levels of variation from block to block in the spike-train analysis: If $w_j^{(c)}$ is small the probability of observing a particular type of neuron doesn't change significantly from block $j - 1$ to $j$, while if $w_j^{(c)} \approx 1$ the mixture probabilities can change quickly (*e.g.*, due to the "appearance"/"disappearance" of a neuron); for $w_j^{(c)}$ in between these extremes, the probability of observing a particular neuron changes slowly/smoothly with consecutive blocks. The model therefore allows a significant degree of flexibility and adaptivity to changes in neuron statistics.

## 2.3 Accounting for refractory time and drift

To demonstrate how one may explicitly account for refractory-time conditions within the model, assume the time difference between spikes $x_{j\nu}^{(c)}$ and $x_{j\nu'}^{(c)}$ is less than the refractory time, while all other spikes have temporal separations greater than the refractory time; we consider two spikes of this type for notational convenience, but the basic formulation below may be readily extended to more than two spikes of this type. We wish to impose that $x_{j\nu}^{(c)}$ and $x_{j\nu'}^{(c)}$ should not be associated with the same cluster/neuron, but otherwise the model is unchanged. Hence, for $n \neq \nu'$, $\theta_{jn}^{(c)} \sim \hat{G}_j^{(c)} = w_j^{(c)} G_j^{(c)} + (1 - w_j^{(c)}) G_{j-1}^{(c)}$ as in (4). Assuming $\hat{G}_j^{(c)} = \sum_{i=1}^{\infty} \hat{\pi}_{ji}^{(c)} \delta_{\theta_i^*}$, we have the new *conditional* generative construction

$$\theta_{j\nu'}^{(c)} | \theta_{j\nu}^{(c)} \sim \sum_{i=1}^{\infty} \frac{\hat{\pi}_{ji}^{(c)} [1 - I(\theta_{j\nu}^{(c)} = \theta_i^*)]}{\sum_{l=1}^{\infty} \hat{\pi}_{jl}^{(c)} [1 - I(\theta_{j\nu}^{(c)} = \theta_l^*)]} \delta_{\theta_i^*} \tag{6}$$

where $I(\cdot)$ is the indicator function (it is equal to one if the argument is true, and it is zero otherwise). This construction imposes that $\theta_{j\nu'}^{(c)} \neq \theta_{j\nu}^{(c)}$, but otherwise the model preserves that the elements of $\{\theta_i^*\}$ are drawn with a relative probability consistent with $\hat{G}_j^{(c)}$. Note that the time associated with a given spike is assumed known after detection (*i.e.*, it is a covariate), and therefore it is known *a priori* for which spikes the above adjustments must be made to the model.

The representation in (6) constitutes a proper generative construction for $\{\theta_{jn}^{(c)}\}$ in the presence of spikes that co-occur within the refractory time, but it complicates inference. Specifically, recall that $G_j^{(c)} = \sum_{i=1}^{\infty} \pi_{ji}^{(c)} \delta_{\theta_i^*}$, with $\pi_{ji}^{(c)} = U_{ji}^{(c)} \prod_{h<i} (1 - U_{jh}^{(c)})$, with $U_{ji}^{(c)} \sim \text{Beta}(1, \alpha_{jc})$. In the original construction, (4) and (5), in which refractory-time violations are not account for, the Gibbs update equations for $\{U_{ji}^{(c)}\}$ are analytic, due to model conjugacy. However, conjugacy for $\{U_{ji}^{(c)}\}$ is lost with (6), and therefore a Metropolis-Hastings (MH) step is required to draw these random variables with an Markov Chain Monte Carlo (MCMC) analysis. This added complexity is often unnecessary, since the number of refractory-time events is typically very small relative to the total number of spikes that must be sorted. Hence, we have successfully implemented the following *approximation* to the above construction. While the $\theta_{j\nu'}^{(c)}$ is drawn as in (6), assigning $\theta_{j\nu'}^{(c)}$ to one of the members of $\{\theta_i^*\}$ while avoiding a refractory-time violation, the update equations for $\{U_{ji}^{(c)}\}$ are executed as they

would be in (4) and (5), without an MH step. In other words, a construction like (6) is used to assign elements of $\{\theta_i^*\}$ to spikes, but after this step the update equations for $\{U_{ji}^{(c)}\}$ are implemented as in the original (conjugate) model. This is essentially the same approach employed in [5], but now in terms of a "stick-breaking" rather than CRP construction of the DP (here an dHDP), and like in [5] we have found this to yield encouraging results (*e.g.*, no refractory-time violations, and sorting in good agreement with "truth" when available).

Finally, in [5] the authors considered a "drift" in the atoms associated with the DP, which here would correspond to a drift in the atoms associated with our dHDP. In this construction, rather that drawing the $\theta_i^* \sim G_0$ once as in (5), one may draw $\theta_i^* \sim G_0$ for the *first* block of time, and then a simple Gaussian auto-regressive model is employed to allow the $\{\theta_i^*\}$ drift a small amount between consecutive blocks. Specifically, if $\{\theta_{ji}^*\}$ represents the atoms for block $j$, then $\theta_{j+1,i}^* \sim \mathcal{N}(\theta_{ji}^*, \beta_0^{-1})$, where it is imposed that $\beta_0$ is large. We examined this within the context of the model proposed here, and for the data considered in Section 4 this added modeling complexity did not change the results significantly, and therefore we did not consider this added complexity when presenting results. This observed un-importance in imposing drift in $\{\theta_{ji}^*\}$ is likely due to the fact that we draw $s_{jn}^{(c)} \sim f(\theta_{jn}^{(c)})$ with a Gaussian $f(\cdot)$, and therefore even if the $\{\theta_{ji}^*\}$ do not change across data blocks, the model allows drift via variations in the draws from the Gaussian (effecting the inferred variance thereof).

## 3 Inference and Computations

For online sorting of spikes, a Chinese restaurant process (CRP) formulation like that in [5] is desirable. The proposed model may be implemented as a generalization of the CRP, as the general form of the model in Section 2.2 is independent of the specific way inference is performed. In a CRP construction, the Chinese restaurant *franchise* (CRF) model [20] is invoked, and the model in Section 2.2 yields a *dynamic* CRF (dCRF), where each franchise is associated with a particular channel. The hierarchical form of the dCRF, including the dictionary-learning component of Section 2.1, is fully conjugate, and may therefore be implemented via a Gibbs sampler.

As hinted by the construction in (6), we here employ a *stick-breaking* construction of the model, analogous to the form of inference employed in [17]. We employ a *retrospective* stick-breaking construction [15] for $G_j^{(c)}$ and $G$ [10], such that the number of terms used to construct $G$ and $G_j^{(c)}$ is unbounded and adapts to the data. Using this construction the model is able to adapt to the number of neurons present, adding and deleting clusters as needed. In this sense the stick-breaking construction may also be considered for online implementations. Further, in this model the parameter Gibbs sampling follows an online-style inference, since the data blocks come in sequentially and the parameters for each block only depend on the previous one or a new component. Therefore, while online implementation is not our principal focus here, it may be executed with the proposed model. We also implemented a CRF implementation, for which there is no truncation. Both inference methods (stick-breaking and CRF implementations) gave very similar results.

Although this paper is not principally focused on online implementations, in the context of such, one may also consider online and evolving learning of the dictionary $\mathbf{D}$ [13]. There is recent research on online dictionary learning, which may be adapted here, using recent extensions via Bayesian formalisms [9]; this would, for example, allow the linear subspace in which the spike shapes reside to adapt/change with data block.

## 4 Example Results

For these experiments we used a truncation level of $K = 60$ dictionary elements. In dictionary learning, the hyperparameters in the gamma priors of $\gamma_c$ and $\eta_p^{(c)}$ were set as $a_{\gamma_c} = 10^{-6}$ and $b_{\gamma_c} = 10^{-6}$, $a_{\eta_p^{(c)}} = 0.1$ and $b_{\eta_p^{(c)}} = 10^{-5}$. In the HDP, we set Ga(1,1) for $\alpha_0$ and $\alpha_c$. In dHDP, we set Ga(1,1) for $\alpha_0$ and $\alpha_{jc}$. Meanwhile, in order to encourage the groups to be shared, we set the prior $\prod_{c=1}^{C} \prod_{j=1}^{J-1} \text{Beta}(w_j^{(c)}; a_w, b_w)$ with $a_w = 0.1$ and $b_w = 1$. These parameters have not been optimized, and many analogous settings yield similar results. We used 5000 burn-in samples and 5000 collection samples in the Gibbs sampler, and we choose the collection sample with the

Table 1: *Summary of results on simulated data.*

| Methods | Channel 1 | Channel 2 | Channel 3 | Average |
|---|---|---|---|---|
| K-means | 96.00% | 96.02% | 95.77% | 95.93% |
| GMM | 84.33% | 94.25% | 91.75% | 90.11% |
| K-means with 2 PCs | 96.8% | 96.9% | 96.50% | 96.81% |
| GMM with 2 PCs | 96.83% | 96.98% | 96.92% | 96.91% |
| DP-DL | 97.00% | 96.92% | **97.08%** | 97.00% |
| HDP-DL | **97.39%** | **97.08%** | **97.08%** | **97.18%** |

maximum likelihood when presenting below example clusterings. For the K-means and GMM, we set the cluster level to 3 in the simulated data and to 2 clusters in the real data (see below).

## 4.1 Simulated Data

In neural spike trains it is very difficult to get ground truth information, so for testing and verification we initially consider simulated data with known ground truth. To generate data we draw from the model $\boldsymbol{x}_n^{(c)} \sim \mathcal{N}(\mathbf{D}(\text{diag}(\boldsymbol{\lambda}^{(c)}))\boldsymbol{s}_n^{(c)}, 0.01\mathbf{I}_D)$. We define $\mathbf{D} \in \mathbb{R}^{D \times K}$ and $\boldsymbol{\lambda}^{(c)} \in \mathbb{R}^K$, which constructs our data from $K = 2$ primary dictionary elements of length $D = 40$ in $C = 3$ channels. These dictionary elements are randomly drawn. We vary $\boldsymbol{\lambda}^{(c)}$ from channel to channel, and for each spike, we generate the feature strength according to $p(\boldsymbol{s}_n^{(c)}) = \sum_{i=1}^3 \pi_i \mathcal{N}(\boldsymbol{s}_n^{(c)}|\boldsymbol{\mu}_i^{(c)}, 0.5\mathbf{I}_K)$ with $\boldsymbol{\pi} = [1/3 \ 1/3 \ 1/3]$, which means that there are three neurons across all the channels. We defined $\boldsymbol{\mu}_i^{(c)} \in \mathbb{R}^K$ as the mean in the feature space for each neuron and shift the neuron mean from channel to channel.

For results we associate each cluster with a neuron and determine the percentage of spikes in their correct cluster. The results are shown in Table 1. The combined Dirichlet process and dictionary learning (DP-DL) give similar results to the GMM with 2 principal components (PCs). Because the DP-DL learns the appropriate number of clusters (three) and dictionary elements (two), these models are expected to perform similarly, except that the DP-DL does not require knowledge of the number of dictionary elements and clusters *a priori*. The HDP-DL is allowed to share global clusters and dictionary elements between channels, which improves results as well.

In Figure 2 the sample posteriors show that we peak at the true values of 3 used "global" clusters (at the top layer of the HDP) and 2 used dictionary elements. Additionally, the HDP shares cluster information between channels, which helps the cluster accuracy. In fact, the spikes at the same time will typically be drawn from the same global cluster despite having independent local clusters as seen in the global cluster from each channel in Figure 2(b). Thus, we can determine a global spike at each time point as well as on each channel.

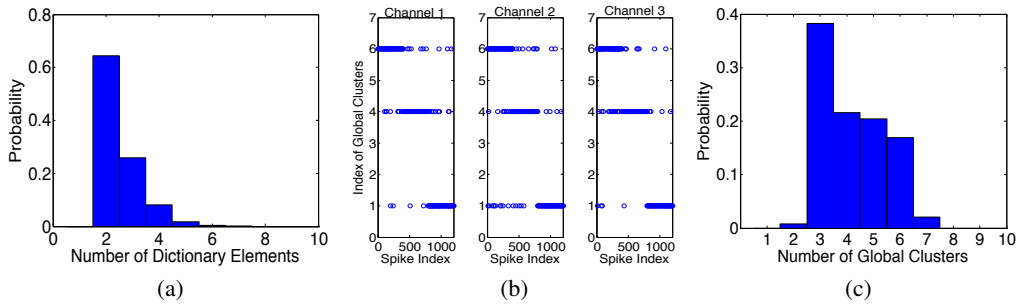

Figure 2: *Posterior information from HDP-DL on simulated data. (a) Approximate posterior distribution of the number of used dictionary elements (i.e., $\|\boldsymbol{b}\|_0$); (b) Example collection sample on the global cluster usage (each local cluster is mapped to its corresponding global index); (c) The approximate posterior distribution on the number of global cluster used.*

Table 2: *Results from testing on d533101 data [7]. KFM represent Kalman Filter Mixture method [2].*

| Methods | Channel 1 | Channel 2 | Channel 3 | Channel 4 | Average |
|---|---|---|---|---|---|
| K-means | 86.67% | 88.04% | 89.20% | 88.4% | 88.08% |
| GMM | 87.43% | 90.06% | 86.75% | 85.43% | 87.42% |
| K-means with 2 PCs | 87.47% | 88.16% | 89.40% | 88.72% | 88.44% |
| GMM with 2 PCs | 89.00% | 89.04% | 87.43% | 90.7% | 89.04% |
| KFM with 2 PCs | 91.00% | 89.2% | 86.35% | 86.87% | 88.36% |
| DP with 2 PCs | 89.04% | 89.00% | 87.43% | 86.79% | 88.07% |
| HDP with 2 PCs | 90.36% | 90.00% | 90.00% | 87.79% | 89.54% |
| DP-DL | 92.29% | 92.38% | 89.52% | 92.45% | 91.89% |
| HDP-DL | **93.38**% | **93.18**% | **93.05**% | **92.61**% | **93.05**% |

## 4.2 Real Data with Partial Ground Truth

We use the publicly available dataset[1] hc-1. These data consist of both extracellular recordings and an intracellular recording from a nearby neuron in the hippocampus of a anesthetized rat [7]. Intracellular recordings give clean signals on a spike train from a specific neuron, giving accurate spike times for that neuron. Thus, if we detect a spike in a nearby extracellular recording within a close time period ($<.5$ms) to an intracellular spike, we assume that the spike detected in the extracellular recording corresponds to the known neuron's spikes. This allows us to know partial ground truth, and allows us to test on methods compared to the known information.

For the accuracy analysis, we determine one cluster that corresponds to the known neuron. Then we consider a spike to be correctly sorted if it is a known spike and is in the known cluster or if it is an unknown spike in the unknown cluster.

In order to give a fair comparison of methods, we first considered the widely used data d533101 and used the same preprocessing from [2]. This data consists of a 4-channel extracellular recordings and 1-channel intracellular recording. We used 2491 detected spikes and 786 of those spikes came from the known neuron. The results are shown in Figure 2. The results show that learning the feature space instead of using the top 2 PCA components increases sorting accuracy. This phenomenon can be seen in Figure 1, where it is impossible to accurately resolve the clusters in the space based on the 2 principal components, through either K-means or GMM. Thus, by jointly learning the suitable feature space and clustering, we are able to separate the unknown and known neurons clusters more accurately. In the HDP model the advantage is clear in the global accuracy as we achieve 89.54% when using 2 PCs and 93.05% when using dictionary learning.

In addition to learning the appropriate feature space, HDP-DL and DP-DL can infer the appropriate number of clusters, allowing the data to define the number of neurons. The posterior distribution on the number of global clusters and number of factors (dictionary elements) used is shown in Figure 3(a) and 3(b), along with the most used elements of the learned dictionary in Figure 3(c). The dictionary elements show shapes similar to both neuron spikes in Figure 3(d) and wavelets. The spiky nature of the learned dictionary can give factors similar to those use in the discrete wavelet transform cluster in [11], which choose to use the Daubechies wavelet for its spiky nature (but here, rather than *a priori* selecting an orthogonal wavelet basis, we *learn* a dictionary that is typically not orthogonal, but is wavelet-like).

Next we used the d561102 data from hc-1, which consists of 4 extracellular recording and 1 intracellular recording. To do spike detection we high-pass filtered the data from 300 Hz and detected spikes when the voltage level passed a positive or negative threshold, as in [2]. We choose this data the known neuron displays dynamic properties by showing periods of activity and inactivity. The intracellular recording in Figure 4(a) shows the known neuron is active for only a brief section of the recorded signal, and is then inactive for the rest of the signal. The nonstationarity passes along to the extracellular spike train and the detect spikes. We used the first 930 detected spikes, which included 202 spikes from the known cluster. In order to model the dynamic properties, we binned the data into 31 subgroups of 30 spikes to use with our multichannel dynamic HDP. The results are shown in

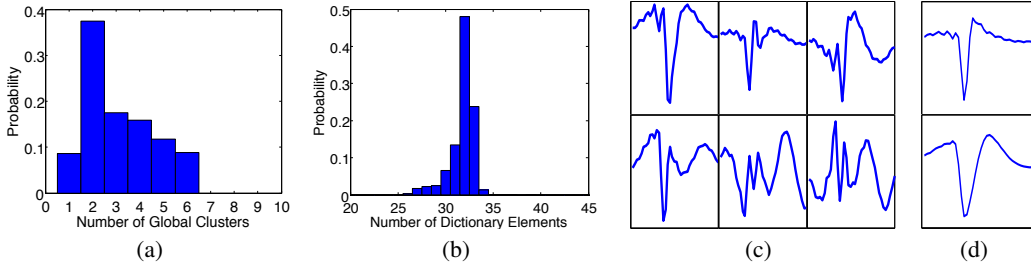

Figure 3: *Results from HDP-DL on d533101 data. (a) approximate posterior probability on the number of global clusters (across all channels); (b) approximate posterior distribution on the number of dictionary elements; (c) six most used dictionary elements; (d) examples of typical spikes from the data.*

Table 3: *Results for d566102 data [7].*

| Methods | Channel 1 | Channel 2 | Channel 3 | Channel 4 | Average |
|---|---|---|---|---|---|
| K-means | 61.82% | 78.77% | 83.59% | 89.39% | 78.39% |
| GMM | 73.85% | 78.66% | 74.18% | 76.59% | 75.82% |
| K-means with 2 PCs | 61.82% | 78.77% | 84.79% | 89.39% | 78.69% |
| GMM with 2 PCs | 75.82% | 78.77% | 75.71% | 88.73% | 79.76% |
| DP-DL | 68.49% | 81.73% | 84.57% | 88.73% | 80.88% |
| HDP-DL | 74.40% | 82.49% | 85.34% | 88.40% | 82.66% |
| MdHDP-DL | **76.04**% | **84.79**% | **87.53**% | **90.48**% | **84.71**% |

Table 3. The model adapts to the nonstationary spike dynamics by learning the parameters to model dynamic properties at block 11 ($w_{11}^{(c)} \approx 1$, indicating that the dHDP has detected a change in the characteristics of the spikes), where the known neuron goes inactive. Thus, the model is more likely to draw new local clusters at this point, reflecting the nonstationary data. Additionally, in Figure 4(c) the global cluster usage shows a dramatic change at time block 11, where a cluster in the model goes inactive at the same time the known neuron is inactive. Because the dynamic model can map these dynamic properties, the results improve while using this model. Additionally, we obtain a global accuracy (across all channels) of 82.66% using the HDP-DL and an global accuracy of 84.71% using the multichannel dynamic HDP-DL (MdHDP-DL). We also tried the KFM on these data, but we were unable to get satisfactory results with it. Additionally, we also calculated the true positive and false positive number to evaluate each method, but due to the limited space, those results were put in Supplementary Material.

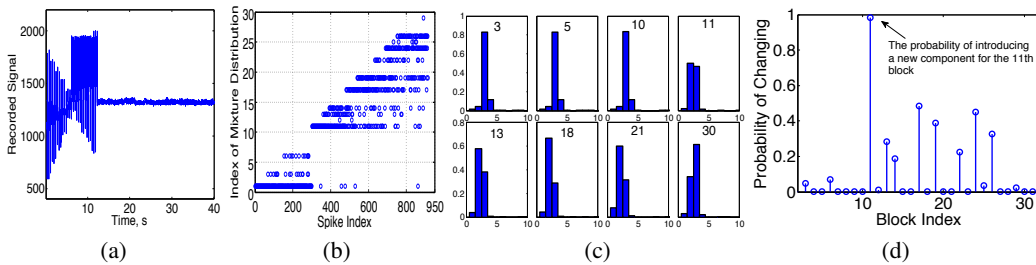

Figure 4: *Results of the multichannel dHDP on d561102. (a) first 40 seconds of the intracellular recording of d561102; (b) local cluster usage by each spike in the d561102 data in channel 4; (c) global cluster usage at different time blocks for the data d561102; (d) sharing weight $w_j^{(c)}$ at each time blocks in the fourth channel. The spike in 11 occurs when the known neuron goes inactive.*

## 5 Conclusions

We have presented a new method for performing multi-channel spike sorting, in which the underlying features (dictionary elements) and sorting are performed jointly, while also allowing time-evolving variation in the spike statistics. The model adaptively learns dictionary elements of a wavelet-like nature (but not orthogonal), with characteristics like the shape of the spikes. Encouraging results have been presented on simulated and real data sets.The authors would like to thank A. Calabrese for providing the KFM codes and processed d533101 data.

## Acknowledgement

The research reported here was supported under the DARPA HIST program.

## Footnotes

[1]available from http://crcns.org/data-sets/hc/hc-1

## References

[1] A. Bar-Hillel, A. Spiro, and E. Stark. Spike sorting: Bayesian clustering of non-stationary data. *J. Neuroscience Methods*, 2006.

[2] A. Calabrese and L. Paniski. Kalman filter mixture model for spike sorting of non-stationary data. *J. Neuroscience Methods*, 2010.

[3] T. S. Ferguson. A Bayesian analysis of some nonparametric problems. *The Annals of Statistics*, 1973.

[4] Y. Gao, M. J. Black, E. Bienenstock, S. Shoham, and J. P. Donoghue. Probabilistic inference of arm motion from neural activity in motor cortex. *Proc. Advances in NIPS*, 2002.

[5] J. Gasthaus, F. Wood, D. Gorur, and Y.W. Teh. Dependent Dirichlet process spike sorting. *In Advances in Neural Information Processing Systems*, 2009.

[6] D. Gorur, C. Rasmussen, A. Tolias, F. Sinz, and N. Logothetis. Modelling spikes with mixtures of factor analysers. *Pattern Recognition*, 2004.

[7] D. A. Henze, Z. Borhegyi, J. Csicsvari, A. Mamiya, K. D. Harris, and G. Buzsaki. Intracellular feautures predicted by extracellular recordings in the hippocampus in vivo. *J. Neurophysiology*, 2010.

[8] J.A. Herbst, S. Gammeter, D. Ferrero, and R.H.R. Hahnloser. Spike sorting with hidden Markov models. *J. Neuroscience Methods*, 2008.

[9] M.D. Hoffman, D.M. Blei, and F. Bach. Online learning for latent Dirichlet allocation. *Proc. NIPS*, 2010.

[10] H. Ishwaran and L.F. James. Gibbs sampling methods for stick-breaking priors. *J. Am. Stat. Ass.*, 2001.

[11] J. C. Letelier and P. P. Weber. Spike sorting based on discrete wavelet transform coefficients. *J. Neuroscience Methods*, 2000.

[12] M. S. Lewicki. A review of methods for spike sorting: the detection and classification of neural action potentials. *Network: Computation in Neural Systems*, 1998.

[13] J. Mairal, F. Bach, J. Ponce, and G. Sapiro. Online learning for matrix factorization and sparse coding. *J. Machine Learning Research*, 2010.

[14] M.A. Nicolelis. Brain-machine interfaces to restore motor function and probe neural circuits. *Nature reviews: Neuroscience*, 2003.

[15] O. Papaspiliopoulos and G. O. Roberts. Retrospective Markov Chain Monte Carlo methods for Dirichlet process hierachiacal models. *Biometrika*, 2008.

[16] C. Pouzat, M. Delescluse, P. Viot, and J. Diebolt. Improved spike-sorting by modeling firing statistics and burst-dependent spike amplitude attenuation: A Markov Chain Monte Carlo approach. *J. Neurophysiology*, 2004.

[17] L. Ren, D. B. Dunson, and L. Carin. The dynamic hierarchical dirichlet process. *International Conference on Machine Learning*, 2008.

[18] G. Santhanam, S.I. Ryu, B.M. Yu, A. Afshar, and K.V. Shenoy. A high-performance brain-computer interface. *Nature*, 2006.

[19] J. Sethuraman. A constructive definition of dirichlet priors. *Statistica Sinica*, 4:639–650, 1994.

[20] Y. W. Teh, M. I. Jordan, M. J. Beal, and D. M. Blei. Hierarchical dirichlet processes. *J. Am. Stat. Ass.*, 2005.

[21] F. Wood, S. Roth, and M. J. Black. Modeling neural population spiking activity with Gibbs distributions. *Proc. Advances in Neural Information Processing Systems*, 2005.

[22] W. Wu, M. J. Black, Y. Gao, E. Bienenstock, M. Serruya, A. Shaikhouni, and J. P. Donoghue. Neural decoding of cursor motion using a Kalman filter. *Proc. Advances in NIPS*, 2003.

